# The CHIR Algorithm for Feed Forward Networks with Binary Weights

**Tal Grossman**
Department of Electronics
Weizmann Institute of Science
Rehovot 76100 Israel

## ABSTRACT

A new learning algorithm, Learning by Choice of Internal Represetations (CHIR), was recently introduced. Whereas many algorithms reduce the learning process to minimizing a cost function over the *weights*, our method treats the *internal representations* as the fundamental entities to be determined. The algorithm applies a search procedure in the space of internal representations, and a cooperative adaptation of the weights (e.g. by using the perceptron learning rule). Since the introduction of its basic, single output version, the CHIR algorithm was generalized to train any feed forward network of binary neurons. Here we present the generalised version of the CHIR algorithm, and further demonstrate its versatility by describing how it can be modified in order to train networks with binary ($\pm 1$) weights. Preliminary tests of this binary version on the random teacher problem are also reported.

## I. INTRODUCTION

Learning by Choice of Internal Representations (CHIR) was recently introduced [1,11] as a training method for feed forward networks of binary units.

Internal Representations are defined as the states taken by the hidden units of a network when patterns (e.g. from the training set) are presented to the input layer of the network. The CHIR algorithm views the internal representations associated with various inputs as the basic independent variables of the learning process. Once such representations are formed, the weights can be found by simple and local learning procedures such as the Perceptron Learning Rule (PLR) [2]. Hence the problem of learning becomes one of *searching for proper internal representations,*

rather than of minimizing a cost function by varying the values of weights, which is the approach used by back propagation (see, however [3],[4] where "back propagation of desired states" is described). This basic idea, of viewing the internal representations as the fundamental entities, has been used since by other groups [5-7]. Some of these works, and the main differences between them and our approach, are briefly disscussed in [11]. One important difference is that the CHIR algorithm, as well as another similar algorithm, the MRII [8], try to solve the learning problem for a fixed architecture, and are not guaranteed to converge. Two other algorithms [5,6] always find a solution, but at the price of increasing the network size during learning in a manner that resembles similar algorithms developed earlier [9,10]. Another approach [7] is to use an error minimizing algorithm which treats the internal representations as well as the weights as the relevant variables of the search space.

To be more specific, consider first the single layer perceptron with its Perceptron Learning Rule (PLR) [2]. This simple network consists of N input (source) units $j$, and a single target unit $i$. This unit is a binary linear threshold unit, i.e. when the source units are set in any one of $\mu = 1, ..M$ patterns, i.e. $S_j = \xi_j^\mu$, the state of unit $i$, $S_i = \pm 1$ is determined according to the rule

$$S_i = sign(\sum_j W_{ij} S_j + \theta_i) \ . \tag{1}$$

Here $W_{ij}$ is the (unidirectional) weight assigned to the connection from unit $j$ to $i$; $\theta_i$ is a local bias. For each of the M input patterns, we require that the target unit (determined using (1)) will take a preassigned value $\xi_i^\mu$. Learning takes place in the course of a training session. Starting from any arbitrary initial guess for the weights, an input $\nu$ is presented, resulting in the output taking some value $S_i^\nu$. Now modify every weight according to the rule

$$W_{ij} \rightarrow W_{ij} + \eta(1 - S_i^\nu \xi_i^\nu)\xi_i^\nu \xi_j^\nu \ , \tag{2}$$

where $\eta > 0$ is a step size parameter ($\xi_j^\nu = 1$ is used to modify the bias $\theta$). Another input pattern is presented, and so on, until all inputs draw the correct output. The Perceptron convergence theorem states [2] that the PLR will find a solution (if one exists), in a finite number of steps. Nevertheless, one needs, for each unit, both the desired input and output states in order to apply the PLR.

Consider now a two layer perceptron, with $N$ input, $H$ hidden and $K$ output units (see Fig.1). The elements of the network are binary linear threshold units $i$, whose states $S_i = \pm 1$ are determined according to (1). In a typical task for such a network, M specified output patterns, $S_i^{out,\mu} = \xi_i^{out,\mu}$, are required in response to $\mu = 1, ..., M$ input patterns. If a solution is found, it first maps each input onto an internal representation generated on the hidden layer, which, in turn, produces the correct output. Now imagine that we are *not* supplied with the weights that solve the problem; however the correct internal representations *are* revealed. That is, we are given a *table* with $M$ rows, one for each input. Every row has $H$ bits $\xi_i^{h,\mu}$, for $i = 1..H$, specifying the state of the hidden layer obtained in response to input

pattern $\mu$. One can now view each hidden-layer cell $i$ as the target of the PLR, with the $N$ inputs viewed as source. Given sufficient time, the PLR will converge to a set of weights $W_{ij}$, connecting input unit $j$ to hidden unit $i$, so that indeed the input-hidden association that appears in column $i$ of our table will be realized. In order to obtain the correct output, we apply the PLR in a learning process that uses the hidden layer as source and each output unit as a target, so as to realize the correct output. In general, however, one is not supplied with a correct table of internal representations. Finding such a table is the goal of our approach.

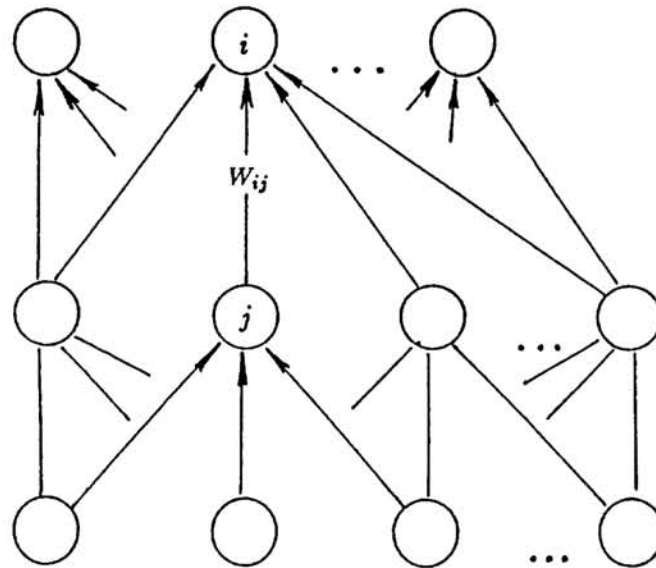

Figure 1. A typical three layered feed forward network (two layered perceptron) with $N$ input, $H$ hidden and $K$ output units. The unidirectional weight $W_{ij}$ connects unit $j$ to unit $i$. A layer index is implicitly included in each unit's index.

During learning, the CHIR algorithm alternates between two phases : in one it generates the internal representations, and in the other it uses the updated representations in order to search for weights, using some single layer learning rule. This general scheme describes a large family of possible algorithms, that use different ways to change the internal representations, and update the weights.

A simple algorithm based on this general scheme was introduced recently [1,11]. In section II we describe the multiple output version of CHIR [11]. In section III we present a way to modify the algorithm so it can train networks with binary weights, and the preliminary results of a few tests done on this new version. In the last section we shortly discuss our results and describe some future directions.

## II. THE CHIR ALGORITHM

The CHIR algorithm that we describe here implements the basic idea of learning by choice of internal representations by breaking the learning process into four distinct procedures that are repeated in a cyclic order :

**1. SETINREP**: Generate a table of internal representations $\{\xi_i^{h,\nu}\}$ by presenting each input pattern from the training set and recording the states of the hidden units, using Eq.(1), with the existing couplings $W_{ij}$ and $\theta_i$.

**2. LEARN23**: The current table of internal representations is used as the training set, the hidden layer cells are used as source, and each output as the target unit of the PLR. If weights $W_{ij}$ and $\theta_i$ that produce the desired outputs are found, the problem has been solved. Otherwise stop after $I_{23}$ learning sweeps, and keep the current weights, to use in CHANGE INREP.

**3. CHANGE INREP**: Generate a new table of internal representations, which reduces the error in the output : We present the table sequentially, row by row (pattern by pattern), to the hidden layer. If for pattern $\nu$ the wrong output is obtained, the internal representation $\xi^{h,\nu}$ is changed.

This is done simply by choosing (at random) a hidden unit $i$, and checking the effect of flipping the sign of $\xi_i^{h,\nu}$ on the total output error, i.e. the number of wrong bits. If the output error is not increased, the flip is accepted and the table of internal representations is changed accordingly. Otherwise the flip is rejected and we try another unit. When we have more than one output unit, it might happen that an error in one output unit can not be corrected without introducing an error in another unit. Therefore we allow only for a pre-specified number of attempted flips, $I_{in}$, and go on to the next pattern even if the output error was not eliminated completely. This procedure ends with a modified, "improved" table which is our next guess of internal representations. Note that this new table does not necessarily yield a totally correct output for all the patterns. In such a case, the learning process will go on even if this new table is perfectly realized by the next stage - LEARN12.

**4. LEARN12**: Present an input pattern; if the output is wrong, apply the PLR with the first layer serving as source, treating every hidden layer site separately as target. If input $\nu$ does yield the correct output, we insert the current state of the hidden layer as the internal representation associated with pattern $\nu$, and no learning steps are taken. We sweep in this manner the training set, modifying weights $W_{ij}$, (between input and hidden layer), hidden-layer thresholds $\theta_i$, and, as explained above, internal representations. If the network has achieved error-free performance for the entire training set, learning is completed. Otherwise, after $I_{12}$ training sweeps (or if the current internal representation is perfectly realized), abort the PLR stage, keeping the present values of $W_{ij}, \theta_i$, and start SETINREP again.

The idea in trying to learn the current internal representation even if it does not yield the perfect output is that it can serve as a better input for the next LEARN23 stage. That way, in each learning cycle the algorithm tries to improve the overall performance of the network.

This algorithm can be further generalized for multi-layered feed forward networks by applying the CHANGE INREP and LEARN12 procedures to each of the hidden layers, one by one, from the last to the first hidden layer.

There are a few details that need to be added.

a) **The "impatience" parameters:** $I_{12}$ and $I_{23}$, which are rather arbitrary, are introduced to guarantee that the PLR stage is aborted if no solution is found, but they have to be large enough to allow the PLR to find a solution (if one exists) with sufficiently high probability. Similar considerations are valid for the $I_{in}$ parameter, the number of flip attempts allowed in the CHANGE INREP procedure. If this number is too small, the updated internal representations may not improve. If it is too large, the new internal representations might be too different from the previous ones, and therefore hard to learn.

The optimal values depend, in general, on the problem and the network size. Our experience indicates, however, that once a "reasonable" range of values is found, performance is fairly insensitive to the precise choice. In addition, a simple rule of thumb can always be applied : "Whenever learning is getting hard, increase the parameters". A detailed study of this issue is reported in [11].

b) **The Internal representations updating scheme:** The CHANGE INREP procedure that is presented here (and studied in [11]) is probably the simplest and "most primitive" way to update the InRep table. The choice of the hidden units to be flipped is completely blind and relies only on the single bit of information about the improvement of the total output error. It may even happen that no change in the internal representaion is made, although such a change is needed. This procedure can certainly be made more efficient, e.g. by probing the fields induced on all the hidden units to be flipped and then choosing one (or more) of them by applying a "minimal disturbance" principle as in [8]. Nevertheless it was shown [11] that even this simple algorithm works quite well.

c) **The weights updating schemes:** In our experiments we have used the simple PLR with a fixed increment ($\eta = 1/2$ , $\Delta W_{ij} = \pm 1$) for weight learning. It has the advantage of allowing the use of discrete (or integer) weights. Nevertheless, it is just a component that can be replaced by other, perhaps more sophisticated methods, in order to achieve, for example, better stability [12], or to take into account various constraints on the weights, e.g. binary weights [13]. In the following section we demonstrate how this can be done.

## III. THE CHIR ALGORITHM FOR BINARY WEIGHTS

In this section we describe how the CHIR algorithm can be used in order to train feed forward networks with binary weights. According to this strong constraint, all the weights in the system (including the thresholds) can be either +1 or -1. The way to do it within the CHIR framework is simple: instead of applying the PLR (or any other single layer, real weights algorithm) for the updating of the weights,

we can use a binary perceptron learning rule. Several ways to solve the learning problem in the binary weight perceptron were suggested recently [13]. The one that we used in the experiments reported here is a modified version of the directed drift algorithm introduced by Venkatesh [13]. Like the standard PLR, the directed drift algorithm works on-line, namely, the patterns are presented one by one, the state of a unit i is calculated according to (1), and whenever an error occurs the incoming weights are updated. When there is an error it means that

$$\xi_i^\nu h_i^\nu < 0$$

Namely, the field $h_i^\nu = \sum_j W_{ij}\xi_j^\nu$ , (induced by the current pattern $\xi_j^\nu$) is "wrong". If so, there must be some weights that pull it to the wrong direction. These are the weights for which

$$\xi_i^\nu W_{ij}\xi_j^\nu < 0.$$

Here $\xi_i^\nu$ is the desired output of unit $i$ for pattern $\nu$. The updating of the weights is done simply by flipping (i.e. $W_{ij} \to -W_{ij}$ ) at random $k$ of these weights.

The number of weights to be changed in each learning step, $k$, can be a pre-fixed parameter of the algorithm, or, as suggested by Venkatesh, can be decreased gradually during the learning process in a way similar to a cooling schedule (as in simulated annealing). What we do is to take $k = |h|/2 + 1$, making sure, like in relaxation algorithms, that just enough weights are flipped in order to obtain the desired target for the current pattern. This simple and local rule is now "plugged" into the Learn12 and Learn23 procedures instead of (2), and the initial weights are chosen to be +1 or -1 at random.

We tested the binary version of CHIR on the "random teacher" problem. In this problem a "teacher network" is created by choosing a random set of +1/-1 weights for the given architecture. The training set is then created by presenting M input patterns to the network and recording the resulting output as the desired output patterns. In what follows we took $M = 2^N$ (exhaustive learning), and an N:N:1 architecture.

The "time" parameter that we use for measuring performance is the number of sweeps through the training set of M patterns ("epochs") needed in order to find the solution. Namely, how many times each pattern was presented to the network. In the experiments presented here, all possible input patterns were presented sequentially in a fixed order (within the perceptron learning sweeps). Therefore in each cycle of the algorithm there are $I_{12} + I_{23} + 1$ such sweeps. Note that according to our definition, a single sweep involves the updating of only one layer of weights or internal representations. for each network size, N, we created an ensemble of 50 independent runs, with different ranodom teachers and starting with a different random choice of initial weights.

We calculate, as a performance measure, the following quantities:

a. The median number of sweeps, $t_m$.

b. The "inverse average rate", $\tau$, as defined by Tesauro and Janssen in [14].

c. The success rate, $S$, i.e. the fraction of runs in which the algorithm finds a solution in less than the maximal number of training cycles $I_{max}$ specified.

The results, with the typical parameters, for N=3,4,5,6, are given in Table 1.

**Table 1.** The Random Teacher problem with N:N:1 architecture.

| N | $I_{12}$ | $I_{23}$ | $I_{in}$ | $I_{max}$ | $t_m$ | $\tau$ | $S$ |
|---|---|---|---|---|---|---|---|
| 3 | 20 | 10 | 5 | 20 | 14 | 9 | 1.00 |
| 4 | 25 | 10 | 7 | 60 | 87 | 37 | 1.00 |
| 5 | 40 | 15 | 9 | 300 | 430 | 60 | 1.00 |
| 6 | 70 | 40 | 11 | 900 | 15000 | 1100 | 0.71 |

As mentioned before, these are only preliminary results. No attempt was made to to optimize the learning parameters.

## IV. DISCUSSION

We presented a generalized version of the CHIR algorithm that is capable of training networks with multiple outputs and hidden layers. A way to modify the basic algortihm so it can be applied to networks with binary weights was also explained and tested. The potential importance of such networks, e.g. in hardware implementation, makes this modified version particularly interesting.

An appealing feature of the CHIR algorithm is the fact that it does not use any kind of "global control", that manipulates the internal representations (as is used for example in [5,6]). The mechanism by which the internal representations are changed is local in the sense that the change is done for each unit and each pattern without conveying any information from other units or patterns (representations). Moreover, the feedback from the "teacher" to the system is only a single bit quantity, namely, whether the output is getting worse or not (in contrast to BP, for example, where one informs each and every output unit about its individual error).

Other advantages of our algorithm are the simplicity of the calculations, the need for only integer, or even binary weights and binary units, and the good performance. It should be mentioned again that the CHIR training sweep involves much less computations than that of back-propagation. The price is the extra memory of $MH$ bits that is needed during the learning process in order to store the internal representations of all $M$ training patterns. This feature is biologically implausible and may be practically limiting. We are developing a method that does not require such memory. The learning method that is currently studied for that purpose [15], is related to the MRII rule, that was recently presented by Widrow and Winter in [8]. It seems that further research will be needed in order to study the practical differences and the relative advantages of the CHIR and the MRII algorithms.

**Acknowledgements :** I am gratefull to Prof. Eytan Domany for many useful suggestions and comments. This research was partially supported by a grant from Minerva.

# References

[1] Grossman T., Meir R. and Domany E., *Complex Systems* **2**, 555 (1989). See also in D. Touretzky (ed.), *Advances in Neural Information Processing Systems 1*, (Morgan Kaufmann, San Mateo 1989).

[2] Minsky M. and Papert S. 1988, *Perceptrons* (MIT, Cambridge);

Rosenblatt F. *Principles of neurodynamics* (Spartan, New York, 1962).

[3] Plaut D.C., Nowlan S.J., and Hinton G.E., Tech.Report CMU-CS-86-126,

Carnegie-Mellon University (1986).

[4] Le Cun Y., *Proc. Cognitiva* **85**, 593 (1985).

[5] Rujan P. and Marchand M., in the *Proc. of the First International Joint Conference Neural Networks - Washington D.C. 1989*, Vol.II, pp. 105. and to appear in *Complex Systems*.

[6] Mezard M. and Nadal J.P., J.Phys.A. **22**, 2191 (1989).

[7] Krogh A., Thorbergsson G.I. and Hertz J.A., in these Proceedings.

R. Rohwer, to apear in the *Proc. of DANIP, GMD Bonn*, April 1989, J. Kinderman and A. Linden eds ;

Saad D. and Merom E., preprint (1989).

[8] Widrow B. and Winter R., *Computer* **21**, No.3, 25 (1988).

[9] See e.g. Cameron S.H., IEEE TEC **EC-13**,299 (1964) ; Hopcroft J.E. and Mattson R.L., IEEE TEC **EC-14**, 552 (1965).

[10] Honavar V. and Uhr L. in the *Proc. of the 1988 Connectionist Models Summer School*, Touretzky D., Hinton G. and Sejnowski T. eds. (Morgan Kaufmann, San Mateo, 1988).

[11] Grossman T., to be published in *Complex Systems* (1990).

[12] Krauth W. and Mezard M., J.Phys.A, **20**, L745 (1988).

[13] Venkatesh S., preprint (1989) ;

Amaldi E. and Nicolis S., J.Phys.France **50**, 2333 (1989).

Kohler H., Diederich S., Kinzel W. and Opper M., preprint (1989).

[14] Tesauro G. and Janssen H., *Complex Systems* **2**, 39 (1988).

[15] Nabutovski D., unpublished.
